# Sparse Signal Recovery Using Markov Random Fields

**Volkan Cevher**
Rice University
volkan@rice.edu

**Marco F. Duarte**
Rice University
duarte@rice.edu

**Chinmay Hegde**
Rice University
chinmay@rice.edu

**Richard G. Baraniuk**
Rice University
richb@rice.edu

## Abstract

Compressive Sensing (CS) combines sampling and compression into a single sub-Nyquist linear measurement process for sparse and compressible signals. In this paper, we extend the theory of CS to include signals that are concisely represented in terms of a *graphical model*. In particular, we use Markov Random Fields (MRFs) to represent sparse signals whose nonzero coefficients are clustered. Our new model-based recovery algorithm, dubbed *Lattice Matching Pursuit* (LaMP), stably recovers MRF-modeled signals using many fewer measurements and computations than the current state-of-the-art algorithms.

## 1 Introduction

The Shannon/Nyquist sampling theorem tells us that in order to preserve information when uniformly sampling a signal we must sample at least two times faster than its bandwidth. In many important and emerging applications, the resulting Nyquist rate can be so high that we end up with too many samples and must compress in order to store or transmit them. In other applications, including imaging systems and high-speed analog-to-digital converters, increasing the sampling rate or density beyond the current state-of-the-art is very expensive. A transform compression system reduces the effective dimensionality of an $N$-dimensional signal by re-representing it in terms of a sparse expansion in some basis (for example, the discrete cosine transform for JPEG). By sparse we mean that only $K \ll N$ of the basis coefficients are nonzero.

The new theory of *compressive sensing* (CS) combines sampling and compression into a single sub-Nyquist linear measurement process for sparse signals [1, 2]. In CS, we measure not periodic signal samples but rather inner products with $M < N$ known measurement vectors; random measurement vectors play a starring role. We then recover the signal by searching for the sparsest signal that agrees with the measurements. Research in CS to date has focused on reducing both the number of measurements $M$ (as a function of $N$ and $K$) and on reducing the computational complexity of the recovery algorithm. Today's state-of-the-art CS systems can recover $K$-sparse and more general compressible signals using $M = O(K \log(N/K))$ measurements using polynomial-time linear programming or greedy algorithms.

While such sub-Nyquist measurement rates are impressive, our contention in this paper is that for CS to truly live up its name it must more fully leverage concepts from state-of-the-art compression algorithms. In virtually all such algorithms, the key ingredient is a *signal model* that goes beyond simple sparsity by providing a model for the basis *coefficient structure*. For instance, JPEG does not only use the fact that most of the DCT of a natural image are small. Rather, it also exploits the fact that the values and locations of the large coefficients have a particular structure that is characteristic of natural images. Coding this structure using an appropriate model enables JPEG and other similar algorithms to compress images close to the maximum amount possible, and significantly better than a naive coder that just assigns bits to each large coefficient independently.

In this paper, we extend the theory of CS to include signals that are concisely represented in terms of a *graphical model* [3]. We use Markov Random Fields (MRFs) to represent sparse signals whose nonzero coefficients also cluster together. Our new model-based recovery algorithm, dubbed *Lattice Matching Pursuit (LaMP)*, performs rapid and numerically stable recovery of MRF-modeled signals using far fewer measurements than standard algorithms.

The organization of the paper is as follows. In Sections 2 and 3, we briefly review the CS and MRF theories. We develop LaMP in Section 4 and present experimental results in Section 5 using both simulated and real world data. We conclude by offering our perspective on the future directions of model-based CS research in Section 6.

## 2 Compressive sensing: From sparsity to *structured* sparsity

**Sparse signal recovery.** Any signal $x \in \mathbb{R}^N$ can be represented in terms of $N$ coefficients $\{\alpha_i\}$ in a basis $\{\psi_i\}_{i=1}^N$; stacking the $\psi_i$ as columns into the matrix $\Psi_{N \times N}$, we can write succinctly that $x = \Psi \theta$. We say that $x$ has a sparse representation if only $K \ll N$ entries of $\theta$ are nonzero, and we denote by $\Omega_K$ the set of $\binom{N}{K}$ possible supports for such $K$-sparse signals. We say that $x$ is *compressible* if the sorted magnitudes of the entries of $\theta$ decay rapidly enough that it can be well approximated as $K$-sparse.

In Compressive Sensing (CS), the signal is not acquired by measuring $x$ or $\alpha$ directly. Rather, we measure the $M < N$ linear projections $y = \Phi x = \Phi \Psi \theta$ using the $M \times N$ matrix $\Phi$. In the sequel, without loss of generality, we focus on two-dimensional image data and assume that $\Psi = I$ (the $N \times N$ identity matrix) so that $x = \theta$. The most commonly used criterion for evaluating the quality of a CS measurement matrix is the restricted isometry property (RIP). A matrix $\Phi$ satisfies the $K$-RIP if there exists a constant $\delta_K > 0$ such that for all $K$-sparse vectors $x$,

$$(1 - \delta_K)\|x\|_2 \leq \|\Phi x\|_2 \leq (1 + \delta_K)\|x\|_2. \tag{1}$$

The recovery of the set of significant coefficients $\theta_i$ is achieved using *optimization*: we search for the sparsest $\theta$ that agrees with the measurements $y$. While in principle recovery is possible using a matrix that has the $2K$-RIP with $\delta_{2K} < 1$, such an optimization is combinatorially complex (NP-complete) and numerically unstable. If we instead use a matrix that has the $3K$-RIP with $\delta_{3K} < 1/2$, then numerically stable recovery is possible in polynomial time using either a linear program [1, 2] or a greedy algorithm [4]. Intriguingly, a random Gaussian or Bernoulli matrix works with high probability, leading to a randomized acquisition protocol instead of uniform sampling.

**Structured sparsity.** While many natural and manmade signals and images can be described to the first-order as sparse or compressible, their sparse supports (set of nonzero coefficients) often have an underlying order. This order plays a central role in the transform compression literature, but it has barely been explored in the CS context [5, 6]. The theme of this paper is that by exploiting a priori information on coefficient structure *in addition* to signal sparsity, we can make CS better, stronger, and faster.

Figure 1 illustrates a real-world example of structured sparse support in a computer vision application. Figure 1(b) is a *background subtracted image* computed from a video sequence of a parking lot with two moving people (one image frame is shown in Figure 1(a)). The moving people form the foreground (white in (b)), while the rest of the scene forms the background (black in (b)). The background subtraction was computed from CS measurements of the video sequence. Background subtracted images play a fundamental role in making inferences about objects and activities in a scene and, by nature, they have structured spatial sparsity corresponding to the foreground innovations. In other words, compared to the scale of the scene, the foreground innovations are usually not only *sparse* but also *clustered* in a distinct way, e.g., corresponding to the silhouettes of humans and vehicles. Nevertheless, this clustering property is not exploited by current CS recovery algorithms.

**Probabilistic RIP.** The RIP treats all possible $K$-sparse supports equally. However, if we incorporate a probabilistic model on our signal supports and consider only the signal supports with the highest likelihoods, then we can potentially do much better in terms of the required number of measurements required for stable recovery.

We say that $\Phi$ satisfies the $(K, \epsilon)$-probabilistic RIP (PRIP) if there exists a constant $\delta_K > 0$ such that for a $K$-sparse signal $x$ generated by a specified probabilistic signal model, (1) holds with probability at least $1 - \epsilon$ over the signal probability space. We propose a preliminary result on the

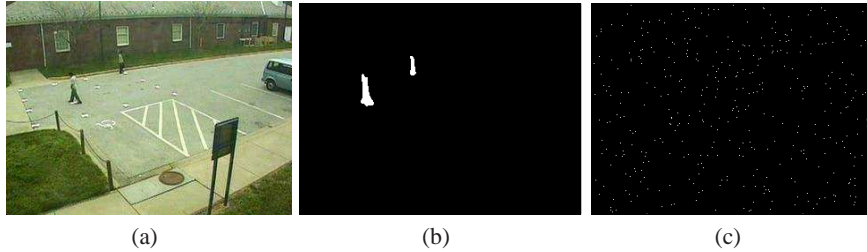

(a)             (b)             (c)

Figure 1: *A camera surveillance image (b) with the background subtracted image (b) recovered using compressive measurements of the scene. The background subtracted image has resolution $N = 240 \times 320$ and sparsity $K = 390$. (c) A random $K = 390$ sparse image in $N = 240 \times 320$ dimensions. The probability of image (b) under the Ising model is approximately $10^{856}$ times greater than the probability of image (c).*

number of random measurements needed under this new criterion; this is a direct consequence of Theorem 5.2 of [8]. (See also [9] for related results.)

**Lemma 1.** *Suppose that $M$, $N$, and $\delta \in [0,1]$ are given and that the signal $\boldsymbol{x}$ is generated by a known probabilistic model. Let $\Omega_{K,\epsilon} \subseteq \Omega_K$ denote the smallest set of supports for which the probability that a $K$-sparse signal $\boldsymbol{x}$ has $\mathrm{supp}(\boldsymbol{x}) \notin \Omega_{K,\epsilon}$ is less than $\epsilon$, and denote $D = |\Omega_{K,\epsilon}|$. If $\boldsymbol{\Phi}$ is a matrix with normalized i.i.d. Gaussian or Bernoulli/Rademacher ($\pm 1$) random entries, then $\boldsymbol{\Phi}$ has the $(K,\epsilon)$-PRIP with probability at least $1 - e^{-c_2 M}$ if $M \geq c_1(K + \log(D))$, where $c_1, c_2 > 0$ depend only on the PRIP constant $\delta_K$.*

To illustrate the significance of the above lemma, consider the following probabilistic model for an $N$-dimensional, $K$-sparse signal. We assume that the locations of the non-zeros follow a homogeneous Poisson process with rate $\lambda = -\log(\epsilon/K)N^{-\alpha}$, where $\alpha \ll 1$. Thus, a particular non-zero coefficient occurs within a distance of $N^\alpha$ of its predecessor with probability $1 - \epsilon/K$. We determine the size of the likely $K$-sparse support set $\Omega_K$ under this particular signal model using a simple counting argument. The location of the first non-zero coefficients is among the first $N^\alpha$ indices with probability $1 - \epsilon/K$. After fixing the location of the first coefficient, the location of the second coefficient is among the next $N^\alpha$ indices immediately following the first location with probability $1 - \epsilon/K$. Proceeding this way, after the locations of the first $j-1$ coefficients, have been fixed, we have that the $j^{\text{th}}$ non-zero coefficient is among $N^\alpha$ candidate locations with probability $1 - \epsilon/K$. In this way, we obtain a set of supports $\Omega_{K,\epsilon}$ of size $N^{\alpha K}$ that will occur with probability $(1 - \epsilon/K)^K > 1 - \epsilon$. Thus for the $(K,\epsilon)$-PRIP to hold for a random matrix, the matrix must have $M = cK(1 + \alpha \log N)$ rows, as compared to the $cK \log(N/K)$ rows required for the standard $K$-RIP to hold. When $\alpha$ is on the order of $(\log N)^{-1}$, the number of measurements required and the complexity of the solution method grow *essentially linearly in* $K$, which is a considerable improvement over the best possible $M = \mathcal{O}(K \log(N/K))$ measurements required without such a priori information.

## 3   Graphical models for compressive sensing

Clustering of the nonzero coefficients in a sparse signal representation can be realistically captured by a probabilistic *graphical model* such as a Markov random field (MRF); in this paper we will focus for concreteness on the classical Ising model [10].

**Support model.** We begin with an Ising model for the signal support. Suppose we have a $K$-sparse signal $\boldsymbol{x} \in \mathbb{R}^N$ whose support is represented by $\boldsymbol{s} \in \{-1,1\}^N$ such that $s_i = -1$ when $x_i = 0$ and $s_i = 1$ when $x_i \neq 0$. The probability density function (PDF) of the signal support can be modeled using a graph $G_s = (V_s, E_s)$, where $V_s = \{1, \ldots, N\}$ denotes a set of $N$ vertices – one for each of the support indices – and $E_s$ denotes the set of edges connecting support indices that are spatial neighbors (see Figure 2(a)). The contribution of the interaction between two elements $\{s_i, s_j\}$ in the support of $\boldsymbol{x}$ is controlled by the coefficient $\lambda_{ij} > 0$. The contribution of each element $s_i$ is controlled by a coefficient $\lambda_i$, resulting in the following PDF for the sparse support $\boldsymbol{s}$:

$$p(\boldsymbol{s}; \boldsymbol{\lambda}) = \exp\left\{ \sum_{(i,j) \in E_s} \lambda_{ij} s_i s_j + \sum_{i \in V_s} \lambda_i s_i - Z_{\boldsymbol{s}}(\boldsymbol{\lambda}) \right\}, \qquad (2)$$

where $Z_{\boldsymbol{s}}(\boldsymbol{\lambda})$ is a strictly convex partition function with respect to $\boldsymbol{\lambda}$ that normalizes the distribution so that it integrates to one. The parameter vector $\boldsymbol{\lambda}$ quantifies our prior knowledge regarding the

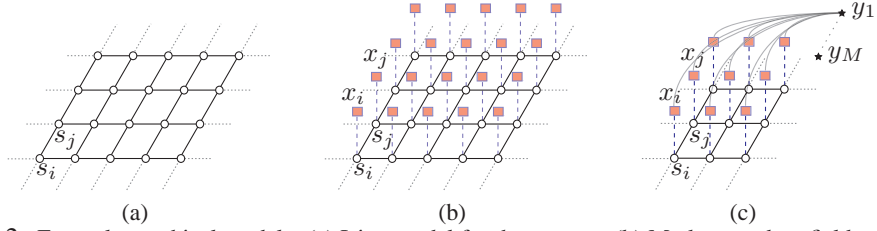

Figure 2: *Example graphical models: (a) Ising model for the support, (b) Markov random field model for the resulting coefficients, (c) Markov random field with CS measurements.*

signal support $s$ and consists of the edge interaction parameters $\lambda_{ij}$ and the vertex bias parameters $\lambda_i$. These parameters can be learned from data using $\ell_1$-minimization techniques [11].

The Ising model enforces coefficient clustering. For example, compare the clustered sparsity of the real background subtracted image in Figure 1(b) with the dispersed "independent" sparsity of the random image in Figure 1(c). While both images (b) and (c) are equally sparse, under a trained Ising model ($\lambda_{ij} = 0.45$ and $\lambda_i = 0$), the image (b) is approximately $10^{856}$ times more likely than the image (c).

**Signal model.** Without loss of generality, we focus on 2D images that are sparse in the space domain, as in Figure 1(b). Leveraging the Ising support model from above, we apply the MRF graphical model in Figure 2(b) for the pixel coefficient values. Under this model, the support is controlled by an Ising model, and the signal values are independent given the support. We now develop a joint PDF for the image pixel values $x$, the support labels $s$, and the CS measurements $y$.

We begin with the support PDF $p(s)$ from (2) and assume that we are equipped with a sparsity-promoting PDF $p(x|s)$ for $x$ given $s$. The most commonly used PDF is the Laplacian density (which is related to the $\ell_1$-norm of $x$); however, other reference priors, such as generalized Gaussians that are related to the $\ell_p$-norm of $x$, $p < 1$, can be more effective [12]. We assume that the measurements $y$ are corrupted by i.i.d. Gaussian noise, i.e., $p(y|x) = \mathcal{N}\left(y|\Phi x, \sigma^2 I\right)$, where $\sigma^2$ is the unknown noise variance.

From Figure 2(c), it is easy to show that, given the signal $x$, the signal support $s$ and the compressive measurements $y$ are independent using the $D$-separation property of graphs [13]. Hence, the joint distribution of the vertices in the graph in Figure 2(b) can be written as

$$p(z) = p(s, x, y) = p(s, x)p(y|s, x) = p(s)p(x|s)p(y|x), \tag{3}$$

where $z = [s^T, x^T, y^T]^T$. Then, (3) can be explicitly written as

$$p(z) \propto \exp\left\{\sum_{(i,j)\in E_s} \lambda_{ij}s_i s_j + \sum_{i\in V_s}[\lambda_i s_i + \log(p(x_i|s_i))] - \frac{1}{2\sigma^2}||y - \Phi x||_2^2\right\}. \tag{4}$$

## 4 Lattice matching pursuit

Using the coefficient graphical model from Section 3, we are now equipped to develop a new model-based CS signal recovery algorithm. Lattice Matching Pursuit (LaMP) is a greedy algorithm for signals on 2D lattices (images) in which the likelihood of the signal support is iteratively evaluated and optimized under an Ising model. By enforcing a graphical model, (*i*) partial knowledge of the sparse signal support greatly decreases the ambiguity and thus size of the search space for the remaining unknown part, accelerating the speed of the algorithm; and (*ii*) signal supports of the same size but different structures result in different likelihoods (recall Figure 1(b) and (c)), decreasing the required number of CS measurements and increasing the numerical stability.

**Algorithm.** The LaMP pseudocode is given in Algorithm 1. Similar to other greedy recovery algorithms such as matching pursuit and CoSaMP [4], each iteration of LaMP starts by estimating a data residual $r^{\{k\}}$ given the current estimate of the signal $x^{\{k-1\}}$ (Step 1). After calculating the residual, LaMP calculates a temporary signal estimate (Step 2) denoted by $x_t^{\{k\}}$. This signal estimate is the sum of the previous estimate $x^{\{k-1\}}$ and $\Phi' r^{\{k\}}$, accounting for the current residual. Using this temporary signal estimate as a starting point, LaMP then maximizes the likelihood (4) over the support via optimization (Step 3). This can be efficiently solved using *graph cuts* with

```
Algorithm 1:  LaMP – Lattice Matching Pursuit
Input: y, Φ, x^{0} = 0, s^{0} = −1, and K̃ (desired sparsity).
Output: A K̃-sparse approximation x of the acquired signal.
Algorithm:
repeat {Matching Pursuit Iterations}
   │  Step 1. Calculate data residual:
   │     r^{k} = y − Φx^{k−1};
   │  Step 2. Propose a temporary target signal estimate:
   │     x_t^{k} = Φ′r^{k} + x^{k−1};
   │  Step 3. Determine MAP estimate of the support using graph cuts:
   │     s^{k} = max_{s∈{−1,+1}^N} Σ_{(i,j)∈E_s} λ_{ij}s_is_j + Σ_{i∈V_s} [λ_is_i + log(p([x_t^{k}]_i|s_i))];
   │  Step 4. Estimate target signal:
   │     t = 0;   t[s^{k} = 1] = Φ^†[:, s^{k} = 1]y;   x^{k} = Prune{t; K̃};
   │  Step 5. Iterate:
   │     k = k + 1;
until Maximum iterations or ‖r^{k}‖ < threshold;
Return  x = x^{k}.
```

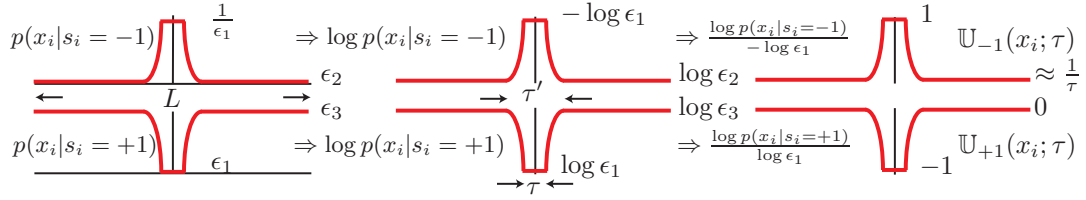

Figure 3: *Geometrical approximations of $p(x_i|s_i = -1)$ and $\log p(x_i|s_i = +1)$.*

$\mathcal{O}(N)$ complexity [14]. In particular, for planar Ising models, the global minimum of the problem can be obtained. Once a likely signal support $s^{\{k\}}$ is obtained in Step 3, LaMP obtains an updated signal estimate $x^{\{k\}}$ using least squares with the selected columns of the measurement matrix $\Phi[:, s^{\{k\}} = 1]$ and pruning back to the largest $\widetilde{K}$ signal coefficients (Step 4). Hence, the parameter $\widetilde{K}$ controls the sparsity of the approximation. In Step 4, a conjugate gradient method is used for efficiently performing the product by a pseudoinverse. If the graphical model includes dependencies between the signal values $x_i$, we then replace the pseudoinverse product by a belief propagation algorithm to efficiently solve for the signal values $x^{\{k\}}$ within Step 4.

**Signal log-likelihood** $\log p(x|s)$. The correct signal PDF to use given the support $p(x|s)$ is problem-dependent. Here, we provide one approximation that mimics the $\ell_0$ minimization for CS recovery for the signal graphical model in Figure 2(c); we also use this in our experiments in Section 5. The state $s_i = 1$ represents a nonzero coefficient; thus, all nonzero values of $x_i$ should have equal probability, and the value $x_i = 0$ should have zero probability. Similarly, the state $s_i = -1$ represents a zero-valued coefficient; thus, the mass of its probability function is concentrated at zero. Hence, we use the approximations for $x_i \in [-L, L]$, a restricted dynamic range: $p(x_i|s_i = -1) = \delta(x_i)$ and $p(x_i|s_i = 1) = (1 - \delta(x_i))/2L$. However, the optimization over the joint PDF in (4) requires a "smoothing" of these PDFs for two reasons: (*i*) to obtain robustness against noise and numerical issues; and (*ii*) to extend the usage of the algorithm from sparse to compressible signals.

We approximate $\log p(x_i|s_i = \pm 1)$ using the parametric form illustrated in Figure 3. Here, the constant $\tau$ is a slack parameter to separate large and small signal coefficients, and $\epsilon_1, \epsilon_2$, and $\epsilon_3$ are chosen according to $\tau$ and $L$ to normalize each PDF. We also denote $a = \epsilon_3 L$, with $a \approx 1$. Using the normalization constraints, it is possible to show that as the dynamic range increases,

$$\lim_{L\to\infty} -\frac{\log \epsilon_2}{\log \epsilon_1} \to \frac{1}{\tau a} \quad \text{and} \quad \lim_{L\to\infty} -\frac{\log \epsilon_3}{\log \epsilon_1} \to 0.$$

Hence, we approximate the likelihoods using the utility functions $\mathbb{U}_{s_i}(x; \tau)$ that follow this form. The optimization problem used by Step 3 of LaMP to determine the support is then approximately equivalent to the following problem

$$s^{\{k+1\}} = \max_{s \in \{-1, +1\}^N} \sum_{(i,j) \in E_s} \widetilde{\lambda}_{ij} s_i s_j + \sum_{i \in V_s} \left[ \widetilde{\lambda}_i s_i + \mathbb{U}_{s_i}([x_t^{\{k+1\}}]_i; \tau) \right], \quad (5)$$

where $\widetilde{\lambda} = \frac{\lambda}{\log \epsilon_1}$. If the signal values are known to be positive, then the definitions of $\mathbb{U}_{s_i}$ can be changed to enforce the positivity during estimation. The choice of $\widetilde{\lambda}_{ij}$ is related to the desired sparseness on the lattice structure.

To enforce a desired sparsity $\widetilde{K}$ on the lattice structure, we apply statistical mechanics results on the 2D Ising model and choose $\widetilde{\lambda}_{ij} = 0.5 \arcsin((1 - m^8)^{-\frac{1}{4}})$, where $m$ is called the average magnetization. In our recovery problem, the average magnetization and the desired signal sparsity has a simple relationship: $m = \left[ (+1) \times \widetilde{K} + (-1) \times (N - \widetilde{K}) \right] / N$. We set $\widetilde{\lambda}_i = 0$ unless there is prior information on the signal support. The threshold $\tau$ is chosen at each iteration adaptively by sorting the magnitudes of the temporary target signal estimate coefficients and determining the $5\widetilde{K}$ threshold; this gives preference to the largest $5\widetilde{K}$ coefficients that attain states $s_i = 1$, unless the cost incurred by enforcing the lattice structure is too large. The pruning operation in Step 4 of LaMP then enforces the desired sparsity $\widetilde{K}$.

## 5 Experiments

We now use several numerical simulations to demonstrate that for spatially clustered sparse signals, which have high likelihood under our MRF model, LaMP requires far fewer measurements and fewer computations for robust signal recovery than state-of-the-art greedy and optimization techniques.[1]

**Experiment 1: Shepp-Logan phantom.** Figure 4 (top left) shows the classical $N = 100 \times 100$ Shepp-Logan phantom image. Its sparsity in the space domain is $K = 1740$. We obtained compressive measurements of this image, which were then immersed in additive white Gaussian noise to an SNR of 10dB. The top row of Figure 4 illustrates the iterative image estimates obtained using LaMP from just $M = 2K = 3480$ random Gaussian measurements of the noisy target. Within 3 iterations, the support of the image is accurately determined; convergence occurs at the 5th iteration.

Figure 4 (bottom) compares LaMP to CoSaMP [4], a state-of-the-art greedy recovery algorithm, and fixed-point continuation (FPC) [17], a state-of-the-art $\ell_1$-norm minimization recovery algorithm using the same set of measurements. Despite the presence of high noise (10dB SNR), LaMP perfectly recovers the signal support from only a small number of measurements. It also outperforms both CoSaMP and FPC in terms of speed.

**Experiment 2: Numerical stability.** We demonstrate LaMP's stability in the face of substantial measurement noise. We tested both LaMP and FPC with a number of measurements that gave close to perfect recovery of the Shepp-Logan phantom in the presence of a small amount of noise; for LaMP, setting $M = 1.7K$ suffices, while FPC requires $M = 4K$. We then studied the degradation of the recovery quality as a function of the noise level for both algorithms. For reference, a value of $\sigma = 20$ corresponds to a measurement-to-noise ratio of just 6dB. The results in Figure 5(a) demonstrate that LaMP is stable for a wide range of measurement noise levels. Indeed, the rate of increase of the LaMP recovery error as a function of the noise variance $\sigma$ (a measure of the stability to noise) is comparable to that of FPC, while using far fewer measurements.

**Experiment 3: Performance on real background subtracted images.** We test the recovery algorithms over a set of background subtraction images. The images were obtained from a test video sequence, one image frame of which is shown in Figure 1, by choosing at random two frames from the video and subtracting them in a pixel-wise fashion. The large-valued pixels in the resulting images are spatially clustered and thus are well-modeled by the MRF enforced by LaMP. We created 100 different test images; for each image, we define the sparsity $K$ as the number of coefficients

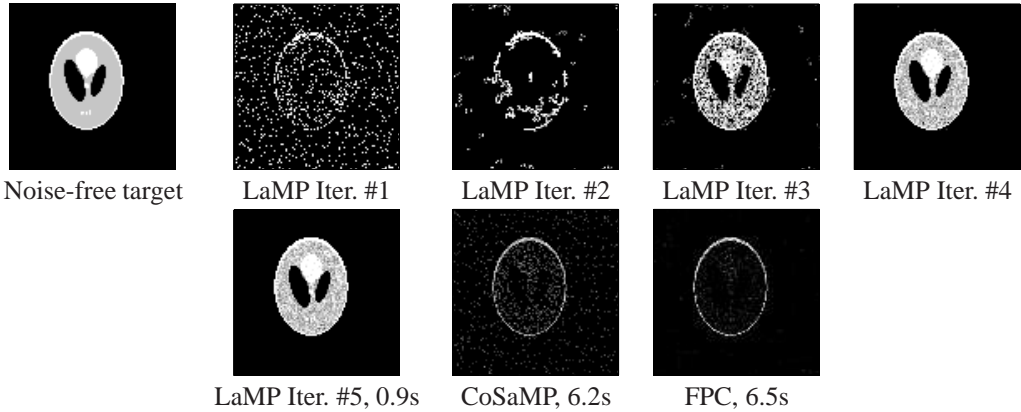

Noise-free target    LaMP Iter. #1    LaMP Iter. #2    LaMP Iter. #3    LaMP Iter. #4

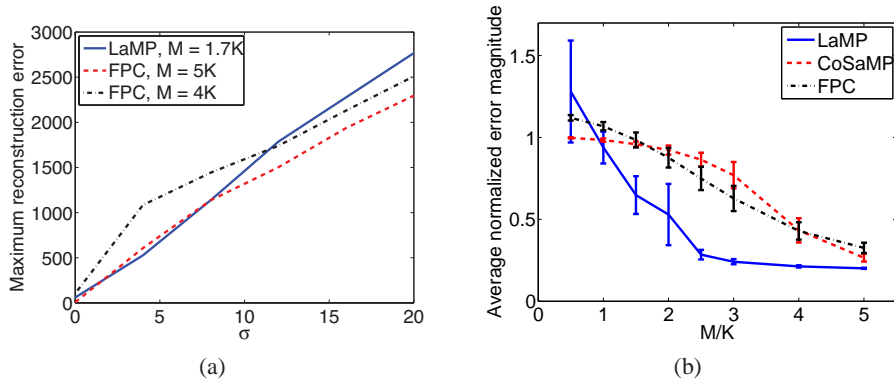

LaMP Iter. #5, 0.9s    CoSaMP, 6.2s    FPC, 6.5s

Figure 4: *Top: LaMP recovery of the Shepp-Logan phantom ($N = 100 \times 100$, $K = 1740$, $\mathrm{SNR} = 10dB$) from $M = 2K = 3480$ noisy measurements. Bottom: Recoveries from LaMP, CoSaMP, and FPC, including running times on the same computer.*

Figure 5: *Performance of LaMP. (a) Maximum recovery error over 1000 noise iterations as a function of the input noise variance. LaMP has the same robustness to noise as the FPC algorithm. (b) Performance over background subtraction dataset of 100 images. LaMP achieves the best performance at $M \approx 2.5K$, while both FPC and CoSaMP require $M > 5K$ to achieve the same performance.*

that contain 97% of the image energy. We then performed recovery of the image using the LaMP, CoSaMP, and FPC algorithms under varying number of measurements $M$, from $0.5K$ to $5K$. An example recovery is shown in Figure 6.

For each test and algorithm, we measured the magnitude of the estimation error normalized by the magnitude of the original image. Figure 5(b) shows the mean and standard deviations for the normalized error magnitudes of the three algorithms. LaMP's graphical model reduces the number of measurements necessary for acceptable recovery quality to $M \approx 2.5K$, while the standard algorithms require $M \geq 5K$ measurements to achieve the same quality.

## 6 Conclusions

We have presented an initial study of model-based CS signal recovery using an MRF model to capture the structure of the signal's sparse coefficients. As demonstrated in our numerical simulations, for signals conforming to our model, the resulting LaMP algorithm requires significantly fewer CS measurements, has lower computational complexity, and has equivalent numerical stability to the current state-of-the-art algorithms. We view this as an initial step toward harnessing the power of modern compression and data modeling methods for CS reconstruction.

Much work needs to be done, however. We are working to precisely quantify the reduction in the required number of measurements (our numerical experiments suggest that $M = O(K)$ is sufficient for stable recovery) and computations. We also assert that probabilistic signal models hold the key to formulating inference problems in the compressive measurement domain since in many signal processing applications, signals are acquired merely for the purpose of making an inference such as a detection or classification decision.

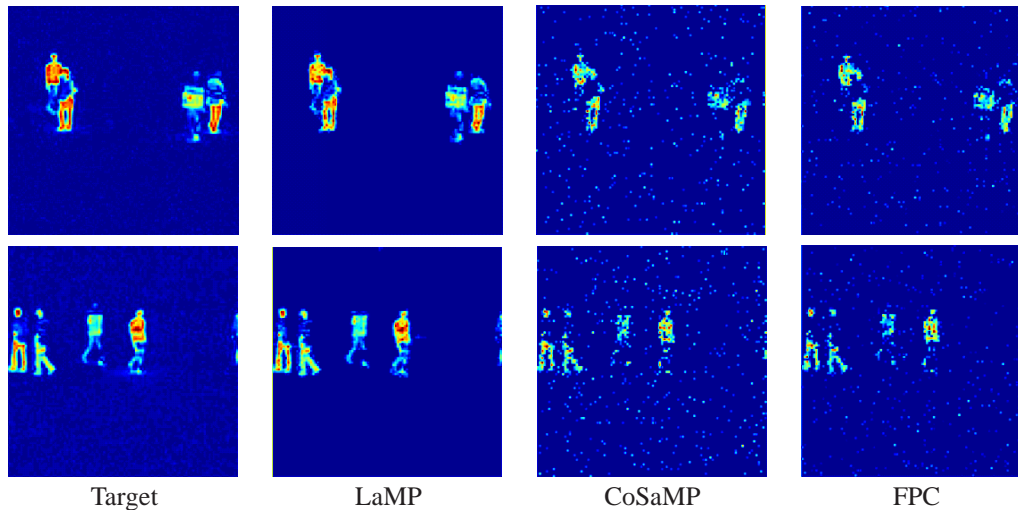

|       |      |        |     |
|-------|------|--------|-----|
| Target | LaMP | CoSaMP | FPC |

Figure 6: *Example recoveries for background subtraction images, using $M = 3K$ for each image.*

**Acknowledgements.** We thank Wotao Yin for helpful discussions, and Aswin Sankaranarayanan for data used in Experiment 3. This work was supported by grants NSF CCF-0431150 and CCF-0728867, DARPA/ONR N66001-08-1-2065, ONR N00014-07-1-0936 and N00014-08-1-1112, AFOSR FA9550-07-1-0301, ARO MURI W311NF-07-1-0185, and the TI Leadership Program.

## Footnotes

[1]We use the *GCOptimization* package [14–16] to solve the support recovery problem in Step 3 in Algorithm 1 in our implementation of LaMP.

# References

[1] D. L. Donoho. Compressed sensing. *IEEE Trans. Info. Theory*, 52(4):1289–1306, Sept. 2006.

[2] E. J. Candès. Compressive sampling. In *Proc. International Congress of Mathematicians*, volume 3, pages 1433–1452, Madrid, Spain, 2006.

[3] S. L. Lauritzen. *Graphical Models*. Oxford University Press, 1996.

[4] D. Needell and J. Tropp. CoSaMP: Iterative signal recovery from incomplete and inaccurate samples. *Applied and Computational Harmonic Analysis*, June 2008. To appear.

[5] C. La and M. N. Do. Tree-based orthogonal matching pursuit algorithm for signal reconstruction. In *IEEE Int. Conf. Image Processing (ICIP)*, pages 1277–1280, Atlanta, GA, Oct. 2006.

[6] M. F. Duarte, M. B. Wakin, and R. G. Baraniuk. Wavelet-domain compressive signal reconstruction using a hidden Markov tree model. In *ICASSP*, pages 5137–5140, Las Vegas, NV, April 2008.

[7] V. Cevher, A. Sankaranarayanan, M. F. Duarte, D. Reddy, R. G. Baraniuk, and R. Chellappa. Compressive sensing for background subtraction. In *ECCV*, Marseille, France, Oct. 2008.

[8] R. G. Baraniuk, M. Davenport, R. A. DeVore, and M. B. Wakin. A simple proof of the restricted isometry property for random matrices. 2006. To appear in *Const. Approx.*

[9] T. Blumensath and M. E. Davies. Sampling theorems for signals from the union of linear subspaces. 2007. Preprint.

[10] B. M. McCoy and T. T. Wu. *The two-dimensional Ising model*. Harvard Univ. Press, 1973.

[11] M. J. Wainwright, P. Ravikumar, and J. D. Lafferty. High-dimensional graphical model selection using $\ell_1$-regularized logistic regression. In *Proc. of Advances in NIPS*, 2006.

[12] D. P. Wipf and B. D. Rao. Sparse bayesian learning for basis selection. *IEEE Trans. Sig. Proc.*, 52(8):2153–2164, August 2004.

[13] J. Pearl. *Probabilistic Reasoning in Intelligent Systems: Networks of Plausible Inference*. Morgan Kaufmann Publishers, 1988.

[14] V. Kolmogorov and R. Zabin. What energy functions can be minimized via graph cuts? *IEEE Trans. on Pattern Anal. and Mach. Int.*, 26(2):147–159, 2004.

[15] Y. Boykov, O. Veksler, and R. Zabih. Efficient approximate energy minimization via graph cuts. *IEEE Trans. on Pattern Anal. and Mach. Int.*, 20(12):1222–1239, Nov. 2001.

[16] Y. Boykov and V. Kolmogorov. An experimental comparison of min-cut/max-flow algorithms for energy minimization in vision. *IEEE Trans. on Pattern Anal. and Mach. Int.*, 26(9):1124–1137, Sept. 2004.

[17] E. T. Hale, W Yin, and Y. Zhang. A fixed-point continuation method for $\ell_1$-regularized minimization with applications to compressed sensing. Technical Report TR07-07, Rice University, CAM Dept., 2007.

